# A Two-Stage Weighting Framework for Multi-Source Domain Adaptation

**Qian Sun**[*]**, Rita Chattopadhyay**[*]**, Sethuraman Panchanathan, Jieping Ye**
Computer Science and Engineering, Arizona State University, AZ 85287
{Qian_Sun, rchattop, panch, Jieping.Ye}@asu.edu

## Abstract

Discriminative learning when training and test data belong to different distributions is a challenging and complex task. Often times we have very few or no labeled data from the test or target distribution but may have plenty of labeled data from multiple related sources with different distributions. The difference in distributions may be both in marginal and conditional probabilities. Most of the existing domain adaptation work focuses on the marginal probability distribution difference between the domains, assuming that the conditional probabilities are similar. However in many real world applications, conditional probability distribution differences are as commonplace as marginal probability differences. In this paper we propose a two-stage domain adaptation methodology which combines weighted data from multiple sources based on marginal probability differences (first stage) as well as conditional probability differences (second stage), with the target domain data. The weights for minimizing the marginal probability differences are estimated independently, while the weights for minimizing conditional probability differences are computed simultaneously by exploiting the potential interaction among multiple sources. We also provide a theoretical analysis on the generalization performance of the proposed multi-source domain adaptation formulation using the weighted Rademacher complexity measure. Empirical comparisons with existing state-of-the-art domain adaptation methods using three real-world datasets demonstrate the effectiveness of the proposed approach.

## 1 Introduction

We consider the domain adaptation scenarios where we have very few or no labeled data from target domain but a large amount of labeled data from multiple related source domains with different data distributions. Under such situations, learning a single or multiple hypotheses on the source domains using traditional machine learning methodologies and applying them on target domain data may lead to poor prediction performance. This is because traditional machine learning algorithms assume that both the source and target domain data are drawn i.i.d. from the same distribution. Figure 1 shows two such source distributions, along with their hypotheses obtained based on traditional machine learning methodologies and a target data distribution. It is evident that the hypotheses learned by the two source distributions D1 and D2 would perform poorly on the target domain data.

One effective approach under such situations is domain adaptation, which enables transfer of knowledge between the source and target domains with dissimilar distributions [1]. It has been applied successfully in various applications including text classification (parts of speech tagging, webpage tagging, etc) [2], video concept detection across different TV channels [3], sentiment analysis (identifying positive and negative reviews across domains) [4] and WiFi Localization (locating device location depending upon the signal strengths from various access points) [5].

---

[*]Authors contributed equally.

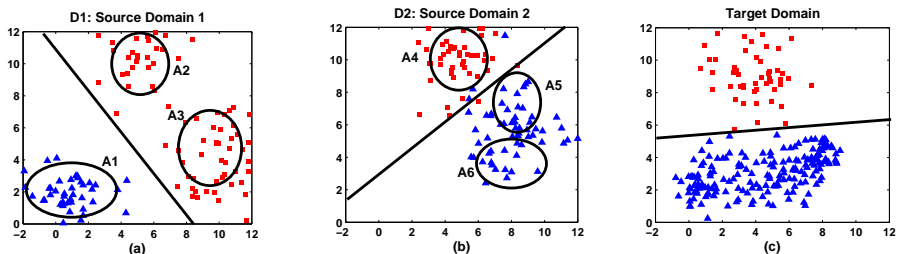

Figure 1: Two source domains D1 and D2 and target domain data with different marginal and conditional probability differences, along with conflicting conditional probabilities (the red squares and blue triangles refer to the positive and negative classes).

Many existing methods re-weight source domain data in order to minimize the marginal probability differences between the source and target domains and learn a hypothesis on the re-weighted source data [6, 7, 8, 9]. However they assume that the distributions differ only in marginal probabilities but the conditional probabilities remain the same. There are other methods that learn model parameters to reduce marginal probability differences [10, 11]. Similarly, several algorithms have been developed in the past to combine knowledge from multiple sources [12, 13, 14]. Most of these methods measure the distribution difference between each source and target domain data, independently, based on marginal or conditional probability differences and combine the hypotheses generated by each of them on the basis of the respective similarity factors. However the example in Figure 1 demonstrates the importance of considering both marginal and conditional probability differences in multi-source domain adaptation.

In this paper we propose a two-stage multi-source domain adaptation framework which computes weights for the data samples from multiple sources to reduce both marginal and conditional probability differences between the source and target domains. In the first stage, we compute weights of the source domain data samples to reduce the marginal probability differences, using Maximum Mean Discrepancy (MMD) [15, 6] as the measure. The second stage computes the weights of multiple sources to reduce the conditional probability differences; the computation is based on the smoothness assumption on the conditional probability distribution of the target domain data [16]. Finally, a target classifier is learned on the re-weighted source domain data. A novel feature of our weighting methodologies is that no labeled data is needed from the target domain, thus widening the scope of their applicability. The proposed framework is readily extendable to the case where a few labeled data may be available from the target domain.

In addition, we present a detailed theoretical analysis on the generalization performance of our proposed framework. The error bound of the proposed target classifier is based on the weighted Rademacher complexity measure of a class of functions or hypotheses, defined over a weighted sample space [17, 18]. The Rademacher complexity measures the ability of a class of functions to fit noise. The empirical Rademacher complexity is data-dependent and can be measured from finite samples. It can lead to tighter bounds than those based on other complexity measures such as the VC-dimension. Theoretical analysis of domain adaptation has been studied in [19, 20]. In [19], the authors provided the generalization bound based on the VC dimension for both single-source and multi-source domain adaptation. The results were extended in [20] to a broader range of prediction problems based on the Rademacher complexity; however only the single-source case was analyzed in [20]. We extend the analysis in [19, 20] to provide the generalization bound for our proposed two-stage framework based on the weighted Rademacher complexity; our generalization bound is tighter than the previous ones in the multi-source case. Our theoretical analysis also reveals the key properties of our generalization bound in terms of a differential weight $\mu$ between the weighted source and target samples.

We have performed extensive experiments using three real-world datasets including 20 Newsgroups, Sentiment Analysis data and one dataset of multi-dimensional feature vectors extracted from Surface Electromyigram (SEMG) signals from eight subjects. SEMG signals are recorded using surface electrodes, from the muscle of a subject, during a submaximal repetitive gripping activity, to detect stages of fatigue. Our empirical results demonstrate superior performance of the proposed approach over the existing state-of-the-art domain adaptation methods; our results also reveal the effect of the differential weight $\mu$ on the target classifier performance.

## 2 Proposed Approach

We consider the following multi-source domain adaptation setting. There are $k$ auxiliary source domains. Each source domain is associated with a sample set $D^s = (x_i^s, y_i^s)|_{i=1}^{n_s}$, $s = 1, 2, \cdots k$, where $x_i^s$ is the $i$-th feature vector, $y_i^s$ is the corresponding class label, $n_s$ is the sample size of the $s$-th source domain, and $k$ is the total number of source domains. The target domain consists of plenty of unlabeled data $D_u^T = x_i^T|_{i=1}^{n_u}$ and optionally a few labeled data $D_l^T = (x_i^T, y_i^T)|_{i=1}^{n_l}$. Here $n_u$ and $n_l$ are the numbers of unlabeled and labeled data, respectively. Denote $D^T = D_l^T \bigcup D_u^T$ and $n_T = n_l + n_u$. The goal is to build a classifier for the target domain data using the source domain data and a few labeled target domain data, if available.

The proposed approach consists of two stages. In the first stage, we compute the weights of source domain data based on the marginal probability difference; in the second stage, we compute the weights of source domains based on the conditional probability difference. A target domain classifier is learned on these re-weighted data.

### 2.1 Re-weighting data samples based on marginal probability differences

The difference between the means of two distributions after mapping onto a reproducing kernel Hilbert space, called Maximum Mean Discrepancy, has been shown to be an effective measure of the differences in their marginal probability distributions [15]. We use this measure to compute the weights $\alpha_i^s$'s of the $s$-th source domain data by solving the following optimization problem [6]:

$$\min_{\alpha^s} \quad \left\| \frac{1}{n_s} \sum_{i=1}^{n_s} \alpha_i^s \Phi(x_i^s) - \frac{1}{n_T} \sum_{i=1}^{n_T} \Phi(x_i^T) \right\|_H^2 \tag{1}$$
$$\text{s.t.} \quad \alpha_i^s \geq 0$$

where $\Phi(x)$ is a feature map onto a reproducing kernel Hilbert space $H$ [21], $n_s$ is the number of samples in the $s$-th source domain, $n_T$ is the number of samples in the target domain, and $\alpha^s$ is the $n_s$ dimensional weight vector. The minimization problem is a standard quadratic problem and can be solved by applying many existing solvers.

### 2.2 Re-weighting Sources based on Conditional probability differences

In the second stage the proposed framework modulates the $\alpha^s$ weights of a source domain $s$ obtained on the basis of marginal probability differences in the first stage, with another weighting factor given by $\beta^s$. The weight $\beta^s$ reflects the similarity of a particular source domain $s$ to the target domain with respect to conditional probability distributions.

Next, we show how to estimate the weights $\beta^s$. For each of the $k$ source domains, a hypothesis $h^s : \mathcal{X} \rightarrow \mathcal{Y}$ is learned on the $\alpha^s$ re-weighted source data samples. This ensures that the hypothesis is learned on source data samples with similar marginal probability distributions. These $k$ source domain hypotheses are used to predict the unlabeled target domain data $D_u^T = x_i^T|_{i=1}^{n_u}$. Let $H_i^S = [h_i^1 \cdots h_i^k]$ be the $1 \times k$ vector of predicted labels of $k$ source domain hypotheses for the $i$-th sample of target domain data. Let $\beta = [\beta^1 \cdots \beta^k]'$ be the $k \times 1$ weight vector, where $\beta^s$ is the weight corresponding to the $s$-th source hypothesis. The estimation of the weight for each source domain hypothesis $h^s$ is based on the smoothness assumption on the conditional probability distribution of the target domain data [16]; specifically we aim to find the optimal weights by minimizing the difference in predicted labels between two nearby points in the target domain as follows.

$$\min_{\beta : \beta' e = 1, \beta \geq 0} \sum_{i,j=1}^{n_u} (H_i^S \beta - H_j^S \beta)^2 W_{ij} \tag{2}$$

where $H^S$ is an $n \times k$ matrix with each row of $H^S$ given by $H_i^S$ as defined above, $H_i^S \beta$ and $H_j^S \beta$ are the predicted labels for the $i$-th and $j$-th samples of target domain data obtained by following a $\beta$ weighted ensemble methodology over all $k$ sources, and $W_{ij}$ is the similarity between the two target domain data samples. We can rewrite the minimization problem as follows:

$$\min_{\beta : \beta' e = 1, \beta \geq 0} \beta' H^{S'} L_u H^S \beta \tag{3}$$

where $L_u$ is the graph Laplacian associated with the target domain data $D_u^T$, given by $L_u = D - W$, where $W$ is the similarity matrix defining edge weights between the data samples in $D_u^T$, and $D$ is the diagonal matrix given by $D_{ii} = \sum_{j=1}^n W_{ij}$. The minimization problem in (3) is a standard quadratic problem (QP) and can be solved efficiently by applying many existing solvers.

To illustrate the proposed two-stage framework, we demonstrate the effect of re-weighting data samples in source domains D1 and D2 of the toy dataset (shown in Figure 1), based on the computed weights, in the supplemental material.

## 2.3 Learning the Target Classifier

The target classifier is learned based on the re-weighted source data and a few labeled target domain data (if available). We also incorporate an additional weighting factor $\mu$ to provide a differential weight to the source domain data with respect to the labeled target domain data. Mathematically, the target classifier $\hat{h}$ is learnt by solving the following optimization problem:

$$\hat{h} = \underset{h}{\mathrm{argmin}} \quad \mu \sum_{s=1}^k \frac{\beta^s}{n_s} \sum_{i=1}^{n_s} \alpha_i^s \mathcal{L}(h(x_i^s), y_i^s) + \sum_{j=1}^{n_l} \frac{1}{n_l} \mathcal{L}(h(x_j^T), y_j^T) \tag{4}$$

where $n_l$ is the number of labeled data from the target domain.

We refer to the proposed framework as *2-Stage Weighting framework for Multi-Source Domain Adaptation (2SW-MDA)*. Algorithm 1 below summarizes the main steps involved in 2SW-MDA.

---
**Algorithm 1** 2SW-MDA
---
1: **for** $s = 1, \ldots, k$ **do**
2:    Compute $\alpha^s$ by solving (1)
3:    Learn a hypothesis $h^s$ on the $\alpha^s$ weighted source data
4: **end for**
5: Form the $n_u \times k$ prediction matrix $H^S$ as in Section 2.2
6: Compute matrices $W$, $D$ and $L$ using the unlabeled target data $D_u^T$
7: Compute $\beta^s$ by solving (3)
8: Learn the target classifier $\hat{h}$ by solving (4)
---

## 3 Theoretical Analysis

For convenience of presentation, we rewrite the empirical joint error function on $(\alpha, \beta)$-weighted source domain and the target domain defined in (4) as follows:

$$\hat{E}_{\alpha,\beta}^S(h) = \mu \hat{\epsilon}_{\alpha,\beta}(h) + \hat{\epsilon}_T(h) = \mu \sum_{s=1}^k \frac{\beta^s}{n_s} \sum_{i=1}^{n_s} \alpha_i^s \mathcal{L}(h(x_i^s), f_s(x_i^s)) + \sum_{i=1}^{n_l} \frac{1}{n_l} \mathcal{L}(h(x_i^0), f_0(x_i^0)) \tag{5}$$

where $y_i^s = f_s(x_i^s)$ and $f_s$ is the labeling function for source $s$, $\mu > 0$, $(x_i^0)$ are samples from the target, $y_i^t = f_0(x_i^0)$ and $f_0$ is the labeling function for the target domain, and $S = (x_i^s)$ include all samples from the target and source domains. The true $(\alpha, \beta)$-weighted error $\epsilon_{\alpha,\beta}(h)$ on weighted source domain samples is defined analogously. Similarly, we define $E_{\alpha,\beta}^S(h)$ as the true joint error function. For notational simplicity, denote $n_0 = n_l$ as the number of labeled samples from the target, $m = \sum_{s=0}^k n_s$ as the total number of samples from both source and target, and $\gamma_s^i = \mu \beta^s \alpha_i^s / n_s$ for $s \geq 1$ and $\gamma_s^i = 1/n$ for $s = 0$. Then we can re-write the empirical joint error function in (5) as:

$$\hat{E}_{\alpha,\beta}^S(h) = \sum_{s=0}^k \sum_{i=1}^{n_s} \gamma_i^s \mathcal{L}(h(x_i^s), f_s(x_i^s)).$$

Next, we bound the difference between the true joint error function $E_{\alpha,\beta}^S(h)$ and its empirical estimate $\hat{E}_{\alpha,\beta}^S(h)$ using the weighted Rademacher complexity measure [17, 18] defined as follows:

**Definition 1.** *(Weighted Rademacher Complexity) Let $\mathbb{H}$ be a set of real-valued functions defined over a set $X$. Given a sample $S \in X^m$, the empirical weighted Rademacher complexity of $\mathbb{H}$ is defined as follows:*

$$\hat{\Re}_S(H) = E_\sigma \left[ \sup_{h \in \mathbb{H}} | \sum_{s=0}^{k} \sum_{i=1}^{n_s} \gamma_i^s \sigma_i^s h(x_i^s)| \, \bigg| \, S = (x_i^s) \right].$$

*The expectation is taken over $\sigma = \{\sigma_i^s\}$ where $\{\sigma_i^s\}$ are independent uniform random variables taking values in $\{-1, +1\}$. The weighted Rademacher complexity of a hypothesis set $\mathbb{H}$ is defined as the expectation of $\hat{\Re}_S(H)$ over all samples of size $m$:*

$$\Re_m(H) = E_S \left[ \hat{\Re}_S(H) \, \big| \, |S| = m \right].$$

Our main result is summarized in the following lemma, which involves the estimation of the Rademacher complexity of the following class of functions:

$$\mathbb{G} = \{x \mapsto \mathcal{L}(h'(x), h(x)) : h, h' \in \mathbb{H}\}.$$

**Lemma 1.** *Let $\mathbb{H}$ be a family of functions taking values in $\{-1, +1\}$. Then, for any $\delta > 0$, with probability at least $1 - \delta$, the following holds for $h \in \mathbb{H}$:*

$$\left| E_{\alpha,\beta}^S(h) - \hat{E}_{\alpha,\beta}^S(h) \right| \le I\!\!R_S(\mathbb{H}) + \sqrt{\frac{\left( \sum_{s=0}^{k} \sum_{i=1}^{n_s} (\gamma_i^s)^2 \right) \log(2/\delta)}{2}}.$$

*Furthermore, if $\mathbb{H}$ has a VC dimension of d, then the following holds with probability at least $1 - \delta$:*

$$\left| E_{\alpha,\beta}^S(h) - \hat{E}_{\alpha,\beta}^S(h) \right| \le \sqrt{\frac{\left( \sum_{s=0}^{k} \sum_{i=1}^{n_s} (\gamma_i^s)^2 \right) \log(2/\delta)}{2}} \left( \sqrt{2d \log \frac{em}{d}} + 1 \right),$$

*where e is the natural number.*

The proof is provided in Section A of the supplemental material.

## 3.1 Error bound on target domain data

In the previous section we presented an upper bound on the difference between the true joint error function and its empirical estimate and established its relation to the weighting factors $\gamma_i^s$. Next we present our main theoretical result, i.e., an upper bound of the error function on target domain data, i.e., an upper bound of $\epsilon_T(\hat{h})$. We need the following definition of divergence for our main result:

**Definition 2.** *For a hypothesis space $\mathcal{H}$, the symmetric difference hypothesis space $d_{\mathbb{H}\Delta\mathbb{H}}$ is the set of hypotheses*

$$g \in \mathbb{H}\Delta\mathbb{H} \Leftrightarrow g(x) = h(x) \oplus h'(x) \; for \; some \; h, h' \in \mathcal{H},$$

*where $\oplus$ is the XOR function. In other words, every hypothesis $g \in \mathbb{H}\Delta\mathbb{H}$ is the set of disagreements between two hypotheses in $\mathcal{H}$.*
*The $\mathbb{H}\Delta\mathbb{H}$-divergence between any two distributions $D_S$ and $D_T$ is defined as*

$$d_{\mathbb{H}\Delta\mathbb{H}}(D_S, D_T)) = 2 \sup_{h,h' \in \mathbb{H}} |Pr_{x \backsim D_S}[h(x) \ne h'(x)] - Pr_{x \backsim D_T}[h(x) \ne h'(x)]|.$$

**Theorem 1.** *Let $\hat{h} \in \mathbb{H}$ be an empirical minimizer of the joint error function on similarity weighted source domain and the target domain:*

$$\hat{h} = arg \min_{h \in \mathbb{H}} \hat{E}_{\alpha,\beta}(h) \equiv \mu \hat{\epsilon}_{\alpha,\beta}(h) + \hat{\epsilon}_T(h)$$

*for fixed weights $\mu$, $\alpha$, and $\beta$ and let $h_T^* = \min_{h \in \mathbb{H}} \epsilon_T(h)$ be a target error minimizer. Then for any $\delta \in (0, 1)$, the following holds with probability at least $1 - \delta$:*

$$\begin{aligned} \epsilon_T(\hat{h}) \quad \le \quad & \epsilon_T(h_T^*) + \frac{2\Re_S(H)}{1+\mu} + \frac{2}{1+\mu} \sqrt{\frac{\left( \sum_{s=0}^{k} \sum_{i=1}^{n_s} (\gamma_i^s)^2 \right) \log(2/\delta)}{2}} \\ & + \frac{\mu}{1+\mu} \left( 2\lambda_{\alpha,\beta} + d_{\mathbb{H}\Delta\mathbb{H}} \left( \mathbb{D}_{\alpha,\beta}, \mathbb{D}_T \right) \right), \end{aligned} \tag{6}$$

*if $\mathbb{H}$ has a VC dimension of $d$, then the following holds with probability at least $1 - \delta$:*

$$
\begin{aligned}
\epsilon_T(\hat{h}) \;\le\; & \; \epsilon_T(h_T^*) + \frac{2}{1+\mu} \left( \sqrt{ \frac{\left( \sum_{s=0}^{k} \sum_{i=1}^{n_s} (\gamma_i^s)^2 \right) \log(2/\delta)}{2} } \left( \sqrt{2d \log \frac{em}{d}} + 1 \right) \right) \\
& + \; \frac{\mu}{1+\mu} \left( 2\lambda_{\alpha,\beta} + d_{\mathbb{H}\Delta\mathbb{H}} \left( \mathbb{D}_{\alpha,\beta}, \mathbb{D}_T \right) \right),
\end{aligned}
\tag{7}
$$

*where $\lambda_{\alpha,\beta} = \min_{h \in \mathbb{H}} \{ \epsilon_T(h) + \epsilon_{\alpha,\beta}(h) \}$, and $d_{\mathbb{H}\Delta\mathbb{H}} \left( \mathbb{D}_{\alpha,\beta}, \mathbb{D}_T \right))$ is the symmetric difference hypothesis space for $(\alpha, \beta)$-weighted source and target domain data.*

The proof as well as a comparison with the result in [19] is provided in the supplemental material.

We observe that $\mu$ and the divergence between the weighted source and target data play significant roles in the generalization bound. Our proposed two-stage weighting scheme aims to reduce the divergence. Next, we analyze the effect of $\mu$. When $\mu = 0$, the bound reduces to the generalization bound using the $n_l$ training samples in the target domain only. As $\mu$ increases, the effect of the source domain data increases. Specifically, when $\mu$ is larger than a certain value, for the bound in (7), as $\mu$ increases, the second term will reduce, while the last term capturing the divergence will increase. In the extreme case when $\mu = \infty$, the second term in (7) can be shown to be the generalization bound using the weighted samples in the source domain only (the target data will not be effective in this case), and the last term equals to $2\lambda_{\alpha,\beta} + d_{\mathbb{H}\Delta\mathbb{H}} \left( \mathbb{D}_{\alpha,\beta}, \mathbb{D}_T \right)$. Thus, effective transfer is possible in this case only if the divergence is small. We also observed in our experiments that the target domain error of the learned joint hypothesis follows a bell shaped curve; it has a different optimal point for each dataset under certain similarity and divergence measures.

## 4 Empirical evaluations

**Datasets.** We evaluate the proposed 2SW-MDA method on three real-world datasets and the toy data shown in Figure 1. The toy dataset is generated using a mixture of Gaussian distributions. It has two classes and three domains, as shown in Figure 1. The two source domains D1 and D2 were created to have both conditional and marginal probability differences with the target domain data so as to provide an ideal testbed for the proposed domain adaptation methodology. The three real-world datasets used are 20 Newsgroups[1], Sentiment Analysis[2] and another dataset of multi-dimensional feature vectors extracted from SEMG (Surface electromyogram) signals. The 20 Newsgroups dataset is a collection of approximately 20,000 newsgroup documents, partitioned (nearly) evenly across 20 different categories. We represented each document as a binary vector of the 100 most discriminating words determined by Weka's info-gain filter [22]. Out of the 20 categories, we used 13 categories, to form the source and target domains. For each of these categories the negative class was formed by a random mixture of the rest of the seven categories, as suggested in [23]. The details of the 13 categories used can be found in the supplemental material. The Sentiment Analysis dataset contains positive and negative reviews on four categories (or domains) including *kitchen*, *book*, *dvd*, and *electronics*. We processed the Sentiment Analysis dataset to reduce the feature dimension to 200 using a cutoff document frequency of 50.

The SEMG dataset is 12-dimensional time and frequency domain features derieved from Surface Electromyogram (SEMG) physiological signals. SEMG are biosignals recorded from the muscle of a subject using surface electrodes to study the muscoskeletal activities of the subject under test. SEMG signals used in our experiments, are recorded from extensor carpi radialis muscle during a submaximal repetitive gripping activity, to study different stages of fatigue. Data is collected from 8 subjects. Each subject data forms a domain. There are 4 classes defining various stages of fatigue. Data from a target subject is classified using the data from the remaining 7 subjects, which form the multiple source domains.

**Competing Methods.** To evaluate the effectiveness of our approach we compare 2SW-MDA with a baseline method SVM-C as well as with five state-of-the-art domain adaptation methods. In SVM-C,

the training data comprises of data from all source domains (12 for 20 Newsgroups data) and the test data is from the remaining one domain as indicated in the first column of the results in Table 1. The recently proposed multi-source domain adaptation methods used for comparison include Locally Weighted Ensemble (LWE) [14] and Domain Adaptation Machine (DAM) [13]. To evaluate the effectiveness of multi-source domain adaption, we also compared with three other state-of-the-art single-source domain adaptation methods, including Kernel Mean Matching (KMM) [6], Transfer Component Analysis (TCA) [11] and Kernel Ensemble (KE) [24].

**Experimental Setup.** Recall that one of the appealing features of the proposed method is that it requires very few or no labeled target domain data. In our experiments, we used only 1 labeled sample per class from the target domain. The results of the proposed 2SW-MDA method are based on $\mu = 1$ (see Figure 2 for results on varying $\mu$). Each experiment was repeated 10 times with random selections of the labeled data. For each experiment, the category shown in first column of Table 1 was used as the target domain and the rest of the categories as the source domains. Different instances of the 20 Newsgroups categories are different random samples of 100 data samples selected from the total 500 data samples in the dataset. Different instances of SEMG dataset are data belonging to different subjects used as target data. Details about the parameter settings are included in the supplemental material.

| Dataset | SVM-C | LWE | KE | KMM | TCA | DAM | 2SW-MDA |
|---|---|---|---|---|---|---|---|
| talk.politics.mideast | 46.00% | 50.66% | 49.01% | 45.78% | 58.66% | 52.03% | 73.49% |
| | 49.33% | 49.39% | 53.48% | 39.75% | 56.00% | 52.00% | 65.06% |
| | 49.33% | 50.27% | 54.67% | 43.37% | 52.04% | 51.81% | 62.65% |
| talk.politics.misc | 48.83% | 53.62% | 46.77% | 62.32% | 55.90% | 53.22% | 63.67% |
| | 48.22% | 51.12% | 48.39% | 59.42% | 53.23% | 54.12% | 60.87% |
| | 48.31% | 50.72% | 55.01% | 59.07% | 54.83% | 54.12% | 68.12% |
| comp.sys.ibm.pc.hardware | 48.42% | 51.25% | 49.50% | 50.56% | 61.25% | 52.50% | 62.92% |
| | 47.44% | 51.44% | 49.44% | 59.55% | 57.50% | 52.50% | 60.67% |
| | 45.93% | 49.88% | 48.00% | 58.43% | 59.75% | 57.80% | 64.04% |
| rec.sport.baseball | 56.25% | 61.51% | 47.50% | 61.79% | 61.75% | 61.25% | 79.78% |
| | 58.75% | 50.09% | 51.25% | 64.04% | 57.75% | 53.75% | 60.22% |
| | 56.35% | 59.26% | 56.25% | 58.43% | 57.83% | 55.05% | 61.24% |
| kitchen | 35.55% | 40.12% | 49.38% | 64.04% | 64.10% | 58.61% | 70.55% |
| electronics | 35.95% | 42.66% | 48.38% | 65.55% | 54.20% | 52.61% | 59.44% |
| book | 37.77% | 40.12% | 49.38% | 58.88% | 55.01% | 54.10% | 59.47% |
| dvd | 36.01% | 49.44% | 48.77% | 50.00% | 50.00% | 50.61% | 51.11% |
| SEMG- 8 subjects | 70.76% | 67.44% | 63.55% | 64.94% | 66.35% | 74.83% | 83.03% |
| | 43.69% | 77.54% | 74.62% | 63.63% | 59.94% | 81.36% | 87.96% |
| | 50.11% | 75.55% | 62.50% | 64.06% | 56.78% | 74.77% | 88.96% |
| | 59.65% | 81.22% | 69.35% | 52.68% | 73.38% | 80.63% | 88.49% |
| | 40.37% | 52.48% | 65.61% | 49.77% | 57.48% | 76.74% | 86.14% |
| | 59.21% | 65.77% | 83.92% | 70.62% | 76.92% | 59.21% | 87.10% |
| | 47.13% | 60.32% | 77.97% | 51.13% | 55.64% | 74.27% | 87.08% |
| | 69.85% | 72.81% | 79.48% | 67.24% | 42.79% | 84.55% | 93.01% |
| Toy data | 60.05% | 75.63% | 81.40% | 68.01% | 64.97% | 84.27% | 98.54% |

Table 1: Comparison of different methods on three real-world and one toy datasets in terms of classification accuracies (%).

**Comparative Studies.** Table 1 shows the classification accuracies of different methods on the real-world and the toy datasets. We observe that SVM-C performs poorly for all cases. This may be attributed to the distribution difference among the multiple source and target domains. We observe that 20 Newsgroups and Sentiment Analysis datasets have predominantly marginal probability differences. In other words, the frequency of a particular word varies from one category of documents to another. In contrary physiological signals, such as SEMG are predominantly different in conditional probability distributions due to the high subject based variability in the power spectrum of these signals and their variations as fatigue sets in [25, 26]. We also observe that the proposed 2SW-MDA method outperforms other domain adaptation methods and achieves higher classification accuracies in most cases, specially for the SEMG dataset. The accuracies of an SVM classifier, on the toy dataset, when learned only on the source domains D1, D2 individually and on the combined source domains, are $64.08\%$ and $71.84\%$ and $60.05\%$ respectively, while 2SW-MDA achieves an accuracy of $98.54\%$. More results are provided in the supplemental material.

It is interesting to note that instance re-weighting method KMM and feature mapping based method TCA, which address marginal probability differences between the source and target domains per-

form better than LWE and KE for both 20 Newsgroups and Sentiment Analysis data. They also perform better than DAM, a multi-source domain adaptation method, based on marginal probability based weighted hypotheses combination. It is worthwhile to note that LWE is based on conditional probability differences and KE tries to address both differences. Thus, it is not surprising that LWE and KE perform better than KMM and TCA for the SEMG dataset, which is predominantly different in conditional probability distributions. DAM too performs better for SEMG signals. However the proposed 2SW-MDA method, which addresses both marginal and conditional probability differences outperforms all the other methods in most cases. Our experiments verify the effectiveness of the proposed two-stage framework.

**Parameter Sensitivity Studies.** In this experiment, we study the effect of $\mu$ on the classification performance. Figure 2 shows the variation in classification accuracies for some cases presented in Table 1, with varying $\mu$ over a range [0 0.001 0.01 0.1 0.3 0.5 1 100 1000]. The x-axis of the figures are in logarithmic scale. The results for the toy data are included in supplemental material. We can observe from the figure that in most cases, the accuracy values increase as $\mu$ increases from 0 to an optimal value and decreases when $\mu$ further increases. When $\mu = 0$ the target classifier is learned only on the few labeled data from the target domain. As $\mu$ increases the transfer of knowledge due to the presence of additional weighted source data has a positive impact leading to increase in classification accuracies in the target domain. We also

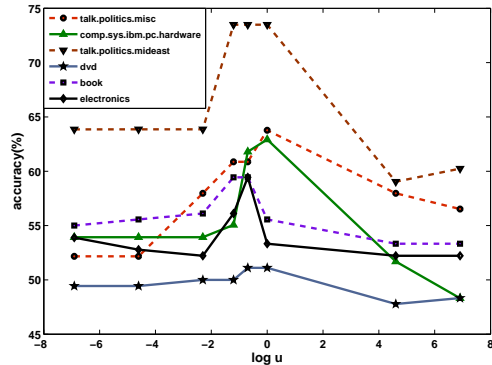

Figure 2: Performance of the proposed 2SW-MDA method on 20 Newsgroups dataset and Sentiment Analysis dataset with varying $\mu$.

observe that after a certain value of $\mu$ the classifier accuracies drop, due to the distribution differences between the source and target domains. These experimental results are consistent with the theoretical results established in this paper.

## 5 Conclusion

Domain adaptation is an important problem that arises in a variety of modern applications where limited or no labeled data is available for a target application. We presented here a novel multi-source domain adaptation framework. The proposed framework computes the weights for the source domain data using a two-step procedure in order to reduce both marginal and conditional probability distribution differences between the source and target domain. We also presented a theoretical error bound on the target classifier learned on re-weighted data samples from multiple sources. Empirical comparisons with existing state-of-the-art domain adaptation methods demonstrate the effectiveness of the proposed approach. As a part of the future work we plan to extend the proposed multi-source framework to applications involving other types of physiological signals for developing generalized models across subjects for emotion and health monitoring [27, 28]. We would also like to extend our framework to video and speech based applications, which are commonly affected by distribution differences [3].

## Acknowledgements

This research is sponsored by NSF IIS-0953662, CCF-1025177, and ONR N00014-11-1-0108.

## Footnotes

[1]Available at http://www.ai.mit.edu/~jrennie/20Newsgroups/

[2]Available at http://www.cs.jhu.edu/~mdredze/

## References

[1] S.J. Pan and Q. Yang. A survey on transfer learning. *IEEE Transactions on Knowledge and Data Engineering*, 2009.

[2] H. Daum III. Frustratingly easy domain adaptation. In *ACL*, 2007.

[3] L. Duan, I.W. Tsang, D. Xu, and S.J. Maybank. Domain transfer svm for video concept detection. In *CVPR*, 2009.

[4] J. Blitzer, M. Dredze, and F. Pereira. Biographies, bollywood, boom-boxes and blenders: Domain adaptation for sentiment classification. In *ACL*, 2007.

[5] S.J. Pan, J.T. Kwok, and Q. Yang. Transfer learning via dimensionality reduction. In *AAAI 08*.

[6] J. Huang, A.J. Smola, A. Gretton, K.M. Borgwardt, and B. Scholkopf. Correcting sample selection bias by unlabeled data. In *NIPS*, volume 19, page 601, 2007.

[7] H. Shimodaira. Improving predictive inference under covariate shift by weighting the log-likelihood function. In *JSPI*, 2000.

[8] S. Bickel, M. Brückner, and T. Scheffer. Discriminative learning under covariate shift. In *JMLR*, 2009.

[9] C. Cortes, Y. Mansour, and M. Mohri. Learning bounds for importance weighing. In *NIPS*, 2010.

[10] M. Sugiyama, S. Nakajima, H. Kashima, P.V. Buenau, and M. Kawanabe. Direct importance estimation with model selection and its application to covariate shift adaptation. In *NIPS*, 2008.

[11] S.J. Pan, I.W. Tsang, J.T. Kwok, and Q. Yang. Domain adaptation via transfer component analysis. In *IJCAI*, 2009.

[12] Y. Mansour, M. Mohri, and A. Rostamizadeh. Domain adaptation with multiple sources. In *NIPS*, 2009.

[13] L. Duan, I.W. Tsang, D. Xu, and T. Chua. Domain adaptation from multiple sources via auxiliary classifiers. In *ICML*, pages 289–296, 2009.

[14] J. Gao, W. Fan, J. Jiang, and J. Han. Knowledge transfer via multiple model local structure mapping. In *KDD*, pages 283–291, 2008.

[15] K.M. Borgwardt, A. Gretton, M.J. Rasch, H.P. Kriegel, B. Scholkopf, and A.J. Smola. Integrating structured biological data by kernel maximum mean discrepancy. In *Bioinformatics*, volume 22, pages 49–57, 2006.

[16] R. Chattopadhyay, J. Ye, S. Panchanathan, W. Fan, and I. Davidson. Multi-source domain adaptation and its application to early detection of fatigue. In *KDD*, 2011.

[17] P.L. Bartlett and S. Mendelson. Rademacher and gaussian complexities: Risk bounds and structural results. *JMLR*, 3:463–482, 2002.

[18] V. Koltchinskii. Rademacher penalties and structural risk minimization. *IEEE Transactions on Information Theory*, 47(5):1902–1914, 2001.

[19] S. Ben-David, J. Blitzer, K. Crammer, A. Kulesza, F. Pereira, and J.W. Vaughan. A theory of learning from different domains. *Journal of Mach Learn*, 79:151–175, 2010.

[20] Y. Mansour, M. Mohri, and A. Rostamizadeh. Domain adaptation: Learning bounds and algorithms. *Computing Research Repository*, abs/0902.3430, 2009.

[21] I. Steinwart. On the influence of the kernel on the consistency of support vector machines. In *JMLR*, volume 2, page 93, 2002.

[22] I.H. Witten and E. Frank. In *Data Mining: Practical Machine Learning Tools with Java Implementations*, San Francisco, CA, 2000. Morgan Kaufmann.

[23] E. Eaton and M. desJardins. Set-based boosting for instance-level transfer. In *IEEE International Conference on Data Mining Workshops*, 2009.

[24] E. Zhong, W. Fan, J. Peng, K. Zhang, J. Ren, D. Turaga, and O. Verscheure. Cross domain distribution adaptation via kernel mapping. In *KDD*, Paris, France, 2009. ACM.

[25] P. Contessa, A. Adam, and C.J. De Luca. Motor unit control and force fluctuation during fatigue. *Journal of Applied Physiology*, April 2009.

[26] B. Gerdle, B. Larsson, and S. Karlsson. Criterion validation of surface EMG variables as fatigue indicators using peak torque: a study of repetitive maximum isokinetic knee extensions. *Journal of Electromyography and Kinesiology*, 10(4):225–232, August 2000.

[27] E. leon, G. Clarke, V. Callaghan, and F. Sepulveda. A user independent real time emotion recognition system for software agents in domestic environment. In *Engineering Application of Artificial Intelligence*, April 2007.

[28] J. Kim and E. Andre. Emotion recognition based on physiological changes in music listening. In *Pattern Analysis and Machine Intelligence*, December 2008.

[29] C. McDiarmid. *On the method of bounded differences.*, volume 5. Cambridge University Press, Cambridge, 1989.

[30] S. Kakade and A. Tewari. Lecture notes of CMSC 35900: Learning theory, Toyota Technological Institute at Chicago. Spring 2008.

[31] P. Massart. Some applications of concentration inequalities to statistics. *Annales de la Faculte des sciences de ToulouseSciences de Toulouse*, IX(2):245–303, 2000.

